# Modeling Conversational Dynamics as a Mixed-Memory Markov Process

**Tanzeem Choudhury**
**Intel Research**
tanzeem.choudhury@intel.com

**Sumit Basu**
**Microsoft Research**
sumitb@microsoft.com

## Abstract

In this work, we quantitatively investigate the ways in which a given person influences the joint turn-taking behavior in a conversation. After collecting an auditory database of social interactions among a group of twenty-three people via wearable sensors (66 hours of data each over two weeks), we apply speech and conversation detection methods to the auditory streams. These methods automatically locate the conversations, determine their participants, and mark which participant was speaking when. We then model the joint turn-taking behavior as a Mixed-Memory Markov Model [1] that combines the statistics of the individual subjects' self-transitions and the partners' cross-transitions. The mixture parameters in this model describe how much each person's individual behavior contributes to the joint turn-taking behavior of the pair. By estimating these parameters, we thus estimate how much influence each participant has in determining the joint turn-taking behavior. We show how this measure correlates significantly with betweenness centrality [2], an independent measure of an individual's importance in a social network. This result suggests that our estimate of conversational influence is predictive of social influence.

## 1    Introduction

People's relationships are largely determined by their social interactions, and the nature of their conversations plays a large part in defining those interactions. There is a long history of work in the social sciences aimed at understanding the interactions between individuals and the influences they have on each others' behavior. However, existing studies of social network interactions have either been restricted to online communities, where unambiguous measurements about how people interact can be obtained, or have been forced to rely on questionnaires or diaries to get data on face-to-face interactions. Survey-based methods are error prone and impractical to scale up. Studies show that self-reports correspond poorly to communication behavior as recorded by independent observers [3].

In contrast, we have used wearable sensors and recent advances in speech processing techniques to automatically gather information about conversations: when they occurred, who was involved, and who was speaking when. Our goal was then to see if we could examine the influence a given speaker had on the turn-taking behavior of her conversational partners. Specifically, we wanted to see if we could better explain the turn-taking transitions observed in a given conversation between subjects $i$ and $j$ by combining the transitions typical to $i$ and those typical to $j$. We could then interpret the contribution from $i$ as her influence on the joint turn-taking behavior.

In this paper, we first describe how we extract speech and conversation information from the raw sensor data, and how we can use this to estimate the underlying social network. We then detail how we use a Mixed-Memory Markov Model to combine the individuals' statistics. Finally, we show the performance of our method on our collected data and how it correlates well with other metrics of social influence.

## 2 Sensing and Modeling Face-to-face Communication Networks

Although people heavily rely on email, telephone, and other virtual means of communication, high complexity information is primarily exchanged through face-to-face interaction [4]. Prior work on sensing face-to-face networks have been based on proximity measures [5],[6], a weak approximation of the actual communication network. Our focus is to model the network based on conversations that take place within a community. To do this, we need to gather data from real-world interactions.

We thus used an experiment conducted at MIT [7] in which 23 people agreed to wear the *sociometer*, a wearable data acquisition board [7],[8]. The device stored audio information from a single microphone at 8 KHz. During the experiment the users wore the device both indoors and outdoors for six hours a day for 11 days. The participants were a mix of students, faculty, and administrative support staff who were distributed across different floors of a laboratory building and across different research groups.

## 3 Speech and Conversation Detection

Given the set of auditory streams of each subject, we now have the problem of detecting who is speaking when and to whom they are speaking. We break this problem into two parts: voicing/speech detection and conversation detection.

### 3.1 Voicing and Speech Detection

To detect the speech, we use the linked-HMM model for voicing and speech detection presented in [9]. This structure models the speech as two layers (see Figure 1); the lower level hidden state represents whether the current frame of audio is voiced or unvoiced (i.e., whether the audio in the frame has a harmonic structure, as in a vowel), while the second level represents whether we are in a speech or non-speech segment. The principle behind the model is that while there are many voiced sounds in our environment (car horns, tones, computer sounds, etc.), the dynamics of voiced/unvoiced transitions provide a unique signature for human speech; the higher level is able to capture this dynamics since the lower level's transitions are dependent on this variable.

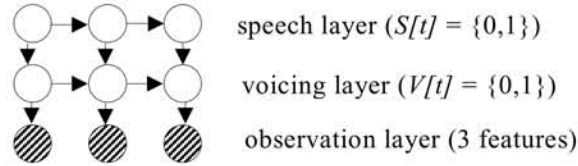

**Figure 1: Graphical model for the voicing and speech detector.**

To apply this model to data, the 8 kHz audio is split into 256-sample frames (32 milliseconds) with a 128-sample overlap. Three features are then computed: the non-initial maximum of the noisy autocorrelation, the number of autocorrelation peaks, and the spectral entropy. The features were modeled as a Gaussian with diagonal covariance. The model was then trained on 8000 frames of fully labeled data. We chose this model because of its robustness to noise and distance from the microphone: even at 20 feet away more than 90% of voiced frames were detected with negligible false alarms (see [9]).

The results from this model are the binary sequences $v[t]$ and $s[t]$ signifying whether the frame is voiced and whether it is in a speech segment for all frames of the audio.

### 3.2 Conversation Detection

Once the voicing and speech segments are identified, we are still left with the problem of determining who was talking with whom and when. To approach this, we use the method of conversation detection described in [10]. The basic idea is simple: since the speech detection method described above is robust to distance, the voicing segments $v[t]$ of all the participants in the conversation will be picked up by the detector in all of the streams (this is referred to as a "mixed stream" in [10]). We can then examine the mutual information of the binary voicing estimates between each person as a matching measure. Since both voicing streams will be nearly identical, the mutual information should peak when the two participants are either involved in a conversation or are overhearing a conversation from a nearby group. However, we have the added complication that the streams are only roughly aligned in time. Thus, we also need to consider a range of time shifts between the streams. We can express the alignment measure $a[k]$ for an offset of $k$ between the two voicing streams as follows:

$$a[k] = I(v_1[t], v_2[t-k]) = \sum_{i,j} p(v_1[t] = i, v_2[t-k] = j) \log \frac{p(v_1[t] = i, v_2[t-l] = j)}{p(v_1[t] = i)p(v_2[t-k] = j)}$$

where $i$ and $j$ take on values $\{0, 1\}$ for unvoiced and voiced states respectively. The distributions for $p(v_1, v_2)$ and its marginals are estimated over a window of one minute ($T$=3750 frames). To see how well this measure performs, we examine an example pair of subjects who had one five-minute conversation over the course of half an hour. The streams are correctly aligned at $k$=0, and by examining the value of $a[k]$ over a large range we can investigate its utility for conversation detection and for aligning the auditory streams (see Figure 2).

The peaks are both strong and unique to the correct alignment ($k$=0), implying that this is indeed a good measure for detecting conversations and aligning the audio in our setup. By choosing the optimal threshold via the ROC curve, we can achieve 100% detection with no false alarms using time windows $T$ of one minute.

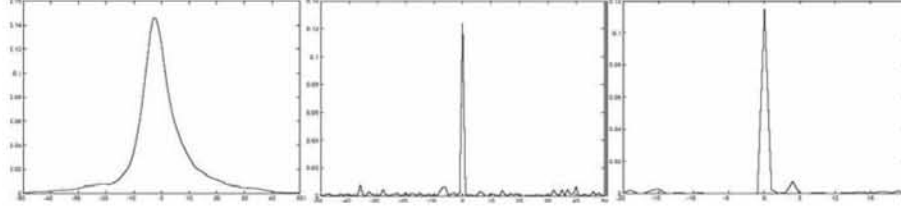

**Figure 2: Values of *a[k]* over ranges: 1.6 seconds, 2.5 minutes, and 11 minutes.**

For each minute of data in each speaker's stream, we computed *a[k]* for *k* ranging over +/- 30 seconds with *T*=3750 for each of the other 22 subjects in the study. While we can now be confident that this will detect most of the conversations between the subjects, since the speech segments from all the participants are being picked up by all of their microphones (and those of others within earshot), there is still the problem of determining who is speaking when. Fortunately, this is fairly straightforward. Since the microphones for each subject are pre-calibrated to have approximately equal energy response, we can classify each voicing segment among the speakers by integrating the audio energy over the segment and choosing the argmax over subjects. It is still possible that the resulting subject does not correspond to the actual speaker (she could simply be the one nearest to a non-subject who is speaking), we determine an overall threshold below which the assignment to the speaker is rejected. Both of these methods are further detailed in [10].

For this work, we rejected all conversations with more than two participants or those that were simply overheard by the subjects. Finally, we tested the overall performance of our method by comparing with a hand-labeling of conversation occurrence and length from four subjects over 2 days (48 hours of data) and found an 87% agreement with the hand labeling. Note that the actual performance may have been better than this, as the labelers did miss some conversations.

### 3.3 The Turn-Taking Signal $S_t^i$

Finally, given the location of the conversations and who is speaking when, we can create a new signal for each subject *i*, $S_t^i$, defined over five-second blocks, which is 1 when the subject is holding the turn and 0 otherwise. We define the holder of the turn as whoever has produced more speech during the five-second block. Thus, within a given conversation between subjects *i* and *j*, the turn-taking signals are complements of each other, i.e., $S_t^i = \neg S_t^j$.

### 4 Estimating the Social Network Structure

Once we have detected the pairwise conversations we can identify the communication that occurs within the community and map the links between individuals. The link structure is calculated from the total number of conversations each subject has with others: interactions with another person that account for less than 5% of the subject's total interactions are removed from the graph. To get an intuitive picture of the interaction pattern within the group, we visualize the network diagram by performing multi-dimensional scaling (MDS) on the geodesic distances (number of hops) between the people (Figure 3). The nodes are colored according to the physical closeness of the subjects' office locations. From this we see that people whose offices are in the same general space seem to be close in the communication space as well.

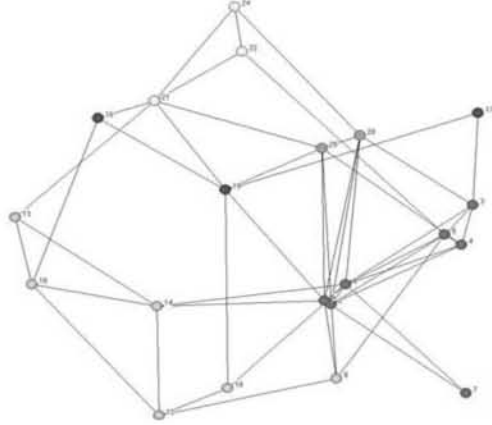

**Figure 3: Estimated network of subjects**

## 5 Modeling the Influence of Turn-taking Behavior in Conversations

When we talk to other people we are influenced by their style of interaction. Sometimes this influence is strong and sometimes insignificant – we are interested in finding a way to quantify this effect. We probably all know people who have a strong effect on our natural interaction style when we talk to them, causing us to change our style as a result. For example, consider someone who never seems to stop talking once it is her turn. She may end up imposing her style on us, and we may consequently end up not having enough of a chance to talk, whereas in most other circumstances we tend to be an active and equal participant.

In our case, we can model this effect via the signals we have already gathered. Let us consider the influence subject $j$ has on subject $i$. We can compute $i$'s average self-transition table, $P(S_t^i \mid S_{t-1}^i)$, via simple counts over all conversations for subject $i$ (excluding those with $j$). Similarly, we can compute $j$'s average cross-transition table, $P(S_t^k \mid S_{t-1}^j)$, over all subjects $k$ (excluding $i$) with which $j$ had conversations. The question now is, for a given conversation between $i$ and $j$, how much does $j$'s average cross-transition help explain $P(S_t^i \mid S_{t-1}^i, S_{t-1}^j)$?

We can formalize this contribution via the Mixed-Memory Markov Model of Saul and Jordan [1]. The basic idea of this model was to approximate a high-dimensional conditional probability table of one variable conditioned on many others as a convex combination of the pairwise conditional tables. For a general set of $N$ interacting Markov chains in the form of a Coupled Markov Model [11], we can write this approximation as:

$$P(S_t^i \mid S_{t-1}^1, ..., S_{t-1}^N) = \sum_j \alpha_{ij} P(S_t^i \mid S_{t-1}^j)$$

For our case of a two chain (two person) model the transition probabilities will be the following:

$$P(S_t^1 \mid S_{t-1}^1, S_{t-1}^2) = \alpha_{11} P(S_t^1 \mid S_{t-1}^1) + \alpha_{12} P(S_t^k \mid S_{t-1}^2)$$

$$P(S_t^2 \mid S_{t-1}^1, S_{t-1}^2) = \alpha_{21} P(S_t^k \mid S_{t-1}^1) + \alpha_{22} P(S_t^2 \mid S_{t-1}^2)$$

This is very similar to the original Mixed-Memory Model, though the transition tables are estimated over all other subjects $k$ excluding the partner as described above. Also, since the $\alpha_{ij}$ sum to one over $j$, in this case $\alpha_{11} = 1 - \alpha_{12}$. We thus have a single parameter, $\alpha_{12}$, which describes the contribution of $P(S_t^k \mid S_{t-1}^2)$ to explaining $P(S_t^1 \mid S_{t-1}^1, S_{t-1}^2)$, i.e., the contribution of subject 2's average turn-taking behavior on her interactions with subject 1.

## 5.1 Learning the influence parameters

To find the $\alpha_{ij}$ values, we would like to maximize the likelihood of the data. Since we have already estimated the relevant conditional probability tables, we can do this via constrained gradient ascent, where we ensure that $\alpha_{ij} > 0$ [12]. Let us first examine how the likelihood function simplifies for the Mixed-Markov model:

$$P(S \mid \{\alpha_{ij}\}) = \left( \prod_i P(S_0^i) \right) \prod_i \prod_t \sum_j \alpha_{ij} P(S_t^i \mid S_{t-1}^j)$$

Converting this expression to log likelihood and removing terms that are not relevant to maximization over $\alpha_{ij}$ yields:

$$\alpha_{ij}^* = \arg\max_{\alpha_{ij}} \left[ \sum_i \sum_t \log \sum_j \alpha_{ij} P(S_t^i \mid S_{t-1}^j) \right]$$

Now we reparametrize for the normality constraint with $\beta_{ij} = \alpha_{ij}$ and $\beta_{iN} = 1 - \sum_j \beta_{ij}$, remove the terms not relevant to chain $i$, and take the derivatives:

$$\frac{\partial}{\partial \beta_{ij}} (.) = \sum_t \frac{P(S_t^i \mid S_{t-1}^j) - P(S_t^i \mid S_{t-1}^N)}{\sum_k \beta_{ik} P(S_t^i \mid S_{t-1}^k) + (1 - \sum_k \beta_{ik}) P(S_t^i \mid S_{t-1}^N)}$$

We can show that the likelihood is convex in the $\alpha_{ij}$, so we are guaranteed to achieve the global maximum by climbing the gradient. More details of this formulation are given in [12],[7].

## 5.2 Aggregate Influence over Multiple Conversations

In order to evaluate whether this model provides additional benefit over using a given subject's self-transition statistics alone, we estimated the reduction in KL divergence by using the mixture of interactions vs. using the self-transition model. We found that by using the mixture model we were able to reduce the KL divergence between a subject's average self-transition statistics and the observed transitions by 32% on average. However, in the mixture model we have added extra degrees of freedom, and hence tested whether the better fit was statistically significant by using the F-test. The resulting p-value was less than 0.01, implying that the mixture model is a significantly better fit to the data.

In order to find a single influence parameter for each person, we took a subset of 80 conversations and aggregated all the pairwise influences each subject had on all her conversational partners. In order to compute this aggregate value, there is an additional aspect about $\alpha_{ij}$ we need to consider. If the subject's self-transition matrix and the complement of the partner's cross-transition matrix are very similar, the influence scores are indeterminate, since for a given interaction $S_t^i = \neg S_t^j$: i.e., we would essentially be trying to find the best way to linearly combine two identical transition matrices. We thus weight the contribution to the aggregate influence estimate for each individual $A_i$ by the relevant J-divergence (symmetrized KL divergence) for each conversational partner:

$$A_i = \sum_{k \in\ partners} J(P(S_t^i \mid \neg S_{t-1}^k) \parallel P(S_t^i \mid S_{t-1}^i))\alpha_{ki}$$

The upper panel of Figure 4 shows the aggregated influence values for the subset of subjects contained in the set of eighty conversations analyzed.

## 6    Link between Conversational Dynamics and Social Role

*Betweenness centrality* is a measure frequently used in social network analysis to characterize importance in the social network. For a given person $i$, it is defined as being proportional to the number of pairs of people $(j,k)$ for which that person lies along the shortest path in the network between $j$ and $k$. It is thus used to estimate how much control an individual has over the interaction of others, since it is a count of how often she is a "gateway" between others. People with high betweenness are often perceived as leaders [2].

We computed the betweenness centrality for the subjects from the 80 conversations using the network structure we estimated in Section 3. We then discovered an interesting and statistically significant correlation between a person's aggregate influence score and her betweenness centrality – it appears that a person's interaction style is indicative of her role within the community based on the centrality measure. Figure 4 shows the weighted influence values along with the centrality scores. Note that ID 8 (the experiment coordinator) is somewhat of an outlier – a plausible explanation for this can be that during the data collection ID 8 went and talked to many of the subjects, which is not her usual behavior. This resulted in her having artificially high centrality (based on link structure) but not high influence based on her interaction style.

We computed the statistical correlation between the influence values and the centrality scores, both including and excluding the outlier subject ID 8. The correlation excluding ID 8 was 0.90 (p-value < 0.0004, rank correlation 0.92) and including ID 8 it was 0.48 (p-value <0.07, rank correlation 0.65). The two measures, namely influence and centrality, are highly correlated, and this correlation is statistically significant when we exclude ID 8, who was the coordinator of the project and whose centrality is likely to be artificially large.

## 7   Conclusion

We have developed a model for quantitatively representing the influence of a given person $j$'s turn-taking behavior on the joint-turn taking behavior with person $i$. On real-world data gathered from wearable sensors, we have estimated the relevant component statistics about turn taking behavior via robust speech processing

techniques, and have shown how we can use the Mixed-Memory Markov formalism to estimate the behavioral influence. Finally, we have shown a strong correlation between a person's aggregate influence value and her betweenness centrality score. This implies that our estimate of conversational influence may be indicative of importance within the social network.

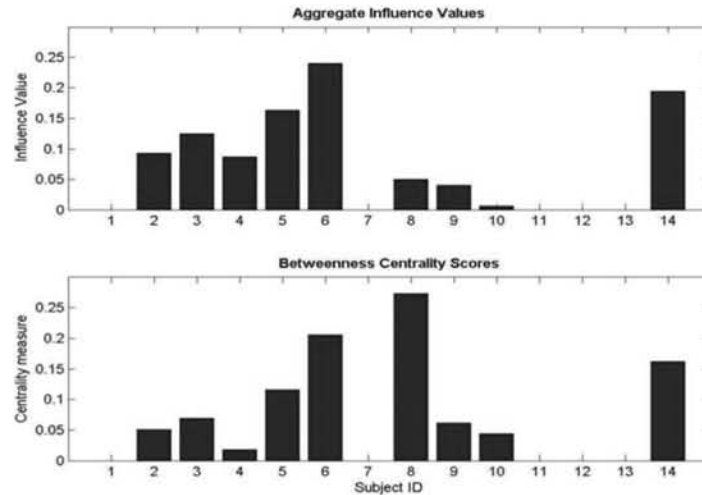

Figure 4: Aggregate influence values and corresponding centrality scores.

## 8   References

[1] Saul, L.K. and M. Jordan. "Mixed Memory Markov Models." *Machine Learning,* 1999. 37: p. 75-85.
[2] Freeman, L.C., "A Set of Measures of Centrality Based on Betweenness." *Sociometry*, 1977. **40**: p. 35-41.
[3] Bernard, H.R., et al., "The Problem of Informant Accuracy: the Validity of Retrospective data." *Annual Review of Anthropology*, 1984. **13**: p. pp. 495-517.
[4] Allen, T., *Architecture and Communication Among Product Development Engineers*. 1997, Sloan School of Management, MIT: Cambridge. p. pp. 1-35.
[5] Want, R., et al., "The Active Badge Location System." *ACM Transactions on Information Systems*, 1992. **10**: p. 91-102.
[6] Borovoy, R., *Folk Computing: Designing Technology to Support Face-to-Face Community Building*. Doctoral Thesis in Media Arts and Sciences. MIT, 2001.
[7] Choudhury, T., Sensing and Modeling Human Networks, Doctoral Thesis in Media Arts and Sciences. MIT. Cambridge, MA, 2003.
[8] Gerasimov, V., T. Selker, and W. Bender, *Sensing and Effecting Environment with Extremity Computing Devices*. Motorola Offspring, 2002. **1**(1).
[9] Basu, S. "A Two-Layer Model for Voicing and Speech Detection." in *Int'l Conference on Acoustics, Speech, and Signal Processing (ICASSP)*. 2003.
[10] Basu, S., *Conversation Scene Analysis*. Doctoral Thesis in Electrical Engineering and Computer Science. MIT. Cambridge, MA 2002.
[11] Brand, M., "Coupled Hidden Markov Models for Modeling Interacting Processes." MIT Media Lab Vision & Modeling Tech Report, 1996.
[12] Basu, S., T. Choudhury, and B. Clarkson. "Learning Human Interactions with the Influence Model." MIT Media Lab Vision and Modeling Tech Report #539. June, 2001.
